# Analytical study of the interplay between architecture and predictability

**Avner Priel, Ido Kanter, David A. Kessler**
Minerva Center and Department of Physics, Bar Ilan University,
Ramat-Gan 52900, Israel.

e-mail: priel@mail.cc.biu.ac.il (web-page: http://faculty.biu.ac.il/~priel)

## Abstract

We study model feed forward networks as time series predictors in the stationary limit. The focus is on complex, yet non-chaotic, behavior. The main question we address is whether the asymptotic behavior is governed by the architecture, regardless the details of the weights. We find hierarchies among classes of architectures with respect to the attractor dimension of the long term sequence they are capable of generating; larger number of hidden units can generate higher dimensional attractors. In the case of a perceptron, we develop the stationary solution for general weights, and show that the flow is typically one dimensional. The relaxation time from an arbitrary initial condition to the stationary solution is found to scale linearly with the size of the network. In multilayer networks, the number of hidden units gives bounds on the number and dimension of the possible attractors. We conclude that long term prediction (in the non-chaotic regime) with such models is governed by attractor dynamics related to the architecture.

Neural networks provide an important tool as model free estimators for the solution of problems when the real model is unknown, or weakly known. In the last decade there has been a growing interest in the application of such tools in the area of time series prediction (see Weigand and Gershenfeld, 1994). In this paper we analyse a typical class of architectures used in this field, i.e. a feed forward network governed by the following dynamic rule:

$$S_1^{t+1} = S_{out} ; \qquad S_j^{t+1} = S_{j-1}^t \quad j = 2, \ldots, N \tag{1}$$

where $S_{out}$ is the network's output at time step $t$ and $S_j^t$ are the inputs at that time; $N$ is the size of the delayed input vector. The rational behind using time delayed vectors as inputs is the theory of state space reconstruction of a dynamic system

using delay coordinates (Takens 1981, Sauer Yorke and Casdagli 1991). This theory address the problem of reproducing a set of states associated with the dynamic system using vectors obtained from the measured time series, and is widely used for time series analysis. A similar architecture incorporating time delays is the TDNN - time-delay neural network with a recurrent loop (Waibel et. al. 1989). This type of networks is known to be appropriate for learning temporal sequences, e.g. speech signal. In the context of time series, it is mostly used for short term predictions. Our analysis focuses on the various long-time properties of the sequence generated by a given architecture and the interplay between them. The aim of such an investigation is the understanding and characterization of the long term sequences generated by such architectures, and the time scale to reach this asymptotic behavior. Such knowledge is necessary to define adequate measures for the transition between a locally dependent prediction and the long term behavior. Though some work has been done on characterization of a dynamic system from its time series using neural networks, not much analytical results that connect architecture and long-time prediction are available (see M. Mozer in Weigand and Gershenfeld, 1994). Nevertheless, practical considerations for choosing the architecture were investigated extensively (Weigand and Gershenfeld, 1994 and references therein). It has been shown that such networks are capable of generating chaotic like sequences. While it is possible to reconstruct approximately the phase space of chaotic attractors (at least in low dimension), it is clear that prediction of chaotic sequences is limited by the very nature of such systems, namely the divergence of the distance between nearby trajectories. Therefore one can only speak about short time predictions with respect to such systems. Our focus is the ability to generate **complex** sequences, and the relation between architecture and the dimension of such sequences.

## 1   Perceptron

We begin with a study of the simplest feed forward network, the perceptron. We analyse a perceptron whose output $S_{out}$ at time step $t$ is given by:

$$S_{out} = \tanh\left[\beta\left(\sum_{j=1}^{N}(W_j + W_0)S_j^t\right)\right] \tag{2}$$

where $\beta$ is a gain parameter, $N$ is the input size. The bias term ,$W_0$, plays the same role as the common 'external field' used in the literature, while preserving the same qualitative asymptotic solution. In a previous work (Eisenstein et. al. , 1995) it was found that the stationary state (of a similar architecture but with a "sign" activation function instead of the "tanh", equivalently $\beta \to \infty$) is influenced primarily by one of the larger Fourier components in the power spectrum of the weights vector $W$ of the perceptron. This observation motivates the following representation of the vector $W$. Let us start with the case of a vector that consists of a single biased Fourier component of the form:

$$W_j = a\cos(2\pi K j/N) \quad j = 1,\ldots,N ; \qquad W_0 = b \tag{3}$$

where $a, b$ are constants and $K$ is a positive integer. This case is generalized later on, however for clarity we treat first the simple case. Note that the vector $W$ can always be represented as a Fourier decomposition of its values. The stationary solution for the sequence $(S^l)$ produced by the output of the perceptron, when inserting this choice of the weights into equation (2), can be shown to be of the form:

$$S^l = \tanh\left[A(\beta)\cos(2\pi K l/N) + B(\beta)\right] \tag{4}$$

There are two non-zero solutions possible for the variables $(A, B)$:

$$A = \tfrac{1}{2}\beta N a \sum_{\rho=1}^{\infty} D(\rho)(A/2)^{2\rho-1}(\rho!)^{-2} \quad ; \quad B = 0$$

$$B = \beta N b \sum_{\rho=1}^{\infty} D(\rho)B^{2\rho-1}((2\rho)!)^{-1} \quad ; \quad A = 0$$

$$(5)$$

where $D(\rho) = 2^{2\rho}(2^{2\rho}-1)\mathcal{B}_{2\rho}$ and $\mathcal{B}_{2\rho}$ are the Bernoulli numbers. Analysis of equations (5) reveals the following behavior as a function of the parameter $\beta$. Each of the variables is the amplitude of an attractor. The attractor represented by $(A \neq 0, B = 0)$ is a limit cycle while the attractor represented by $(B \neq 0, A = 0)$ is a fixed point of the dynamics. The onset of each of the attractors $A(B)$ is at $\beta_{c1} = 2(aN)^{-1}$ $(\beta_{c2} = (bN)^{-1})$ respectively. One can identify three regimes: (1) $\beta < \beta_{c1,c2}$ - the stable solution is $S^l = 0$. (2) $min(\beta_{c1}, \beta_{c2}) < \beta < max(\beta_{c1}, \beta_{c2})$ - the system flows for all initial conditions into the attractor whose $\beta_c$ is smaller. (3) $\beta > \beta_{c1,c2}$ - depending on the initial condition of the input vector, the system flows into one of the attractors, namely, the stationary state is either a fixed point or a periodic flow. $\beta_{c1}$ is known as a Hopf bifurcation point. Naturally, the attractor whose $\beta_c$ is smaller has a larger basin of attraction, hence it is more probable to attract the flow (in the third regime).

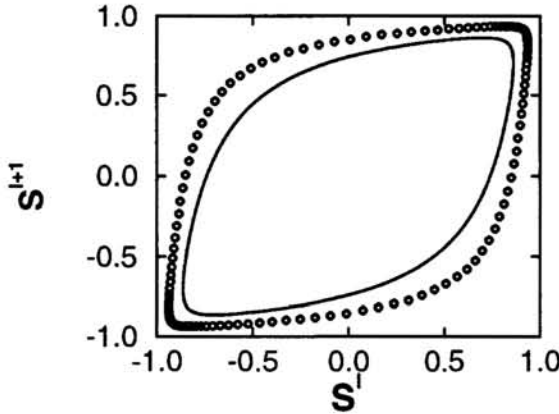

Figure 1: Embedding of a sequence generated by a perceptron whose weights follow eq. 3 (6). Periodic sequence (outer curve) $N = 128$, $k = 17$, $b = 0.3$, $\beta = 1/40$ and quasi periodic (inner) $k = 17$, $\phi = 0.123$, $\beta = 1/45$ respectively.

Next we discuss the more general case where the weights of eq. (3) includes an arbitrary phase shift of the form:

$$W_j = a \cos(2\pi K j/N - \pi\phi) \quad \phi \in (-1,1) \tag{6}$$

The leading term of the stationary solution in the limit $N \gg 1$ is of the form:

$$S^l = \tanh\left[A(\beta)\cos(2\pi(K-\phi)l/N) + B(\beta)\right] \tag{7}$$

where the higher harmonic corrections are of $\mathcal{O}(1/K)$. A note should be made here that the phase shift in the weights is manifested as a frequency shift in the solution. In addition, the attractor associated with $A \neq 0$ is now a quasi-periodic flow in the generic case when $\phi$ is irrational. The onset value of the fixed point $(\beta_{c2})$ is the same as before, however the onset of the quasi-periodic orbit is $\beta_{c1} = \frac{\pi\phi}{\sin(\pi\phi)}2(aN)^{-1}$. The variables $A, B$ follow similar equations to (5):

$$A = \beta N a \frac{\sin(\pi\phi)}{\pi\phi} \sum_{\rho=1}^{\infty} D(\rho)(A/2)^{2\rho-1}(\rho!)^{-2} \quad ; \quad B = 0$$

$$B = \beta N b \sum_{\rho=1}^{\infty} D(\rho)B^{2\rho-1}((2\rho)!)^{-1} \quad ; \quad A = 0$$

$$(8)$$

The three regimes discussed above appear in this case as well. Figure 1 shows the attractor associated with $(A \neq 0, B = 0)$ for the two cases where the series generated by the output is embedded as a sequence of two dimensional vectors $(S^{l+1}, S^l)$.

The general weights can be written as a combination of their Fourier components with different $K$'s and $\phi$'s:

$$W_j = \sum_{i=1}^{m} a_i \cos(2\pi K_i j/N - \pi\phi_i) \quad \phi_i \in (-1,1) \tag{9}$$

When the different $K$'s are not integer divisors of each other, the general solution is similar to that described above:

$$S^l = \tanh\left[\sum_{i=1}^{m} A_i(\beta) \cos(2\pi(K_i - \phi_i)l/N) + B(\beta)\right] \tag{10}$$

where $m$ is the number of relevant Fourier components. As above, the variables $A_i$, $B$ are coupled via self consistent equations. Nevertheless, the generic stationary flow is one of the possible attractors, depending on $\beta$ and the initial condition; i.e. $(A_q \neq 0, A_i = 0 \ \forall i \neq q, B = 0)$ or $(B \neq 0, A_i = 0)$. By now we can conclude that the generic flow for the perceptron is one of three: a fixed point, periodic cycle or quasi-periodic flow. The first two have a zero dimension while the last describes a one dimensional flow. we stress that more complex flows are possible even in our solution (eq. 10), however they require special relation between the frequencies and a very high value of $\beta$, typically more than an order of magnitude greater than bifurcation value.

## 2   Relaxation time

At this stage the reader might wonder about the relation between the asymptotic results presented above and the ability of such a model to predict. In fact, the practical use of feed forward networks in time series prediction is divided into two phases. In the first phase, the network is trained in an open loop using a given time series. In the second phase, the network operates in a closed loop and the sequence it generates is also used for the future predictions. Hence, it is clear from our analysis that eventually the network will be driven to one of the attractors. The relevant question is   *how long does it takes to arrive at such asymptotic behavior ?*   We shall see that the characteristic time is governed by the gap between the largest and the second largest eigenvalues of the linearized map. Let us start by reformulating eqs. (1, 2) in a matrix form, i.e. we linearize the map. Denote $\overline{S}^t = (S_1^t, S_2^t, \ldots, S_N^t)$ and $(\overline{S}^t)'$ is the transposed vector. The map is then   $\mathcal{T}(\overline{S}^t)' = (\overline{S}^{t+1})'$   where

$$\mathcal{T} = \begin{bmatrix} c_1 & c_2 & \cdots & c_{N-1} & c_N \\ 1 & 0 & \cdots & 0 & 0 \\ 0 & 1 & \cdots & 0 & 0 \\ \vdots & \vdots & \vdots & \vdots & \vdots \\ 0 & 0 & \cdots & 1 & 0 \end{bmatrix} \qquad c_i = \beta(W_i + W_0) \tag{11}$$

The first row of $\mathcal{T}$ gives the next output value $= S_1^{t+1}$ while the rest of the matrix is just the shift defined by eq. (1) . This matrix is known as the "companion matrix" (e.g. Ralston and Rabinowitz, 1978). The characteristic function of $\mathcal{T}$ can be written as follows:

$$\beta \sum_{n=1}^{N} \frac{c_n}{\lambda^n} = 1 \tag{12}$$

from which it is possible to extract the eigenvalues. At $\beta = \beta_c$ the largest eigenvalue of $\mathcal{T}$ is $|\lambda_1| = 1$. Denote the second largest eigenvalue $\lambda_2$ such that $|\lambda_2| = 1 - \Delta$ .

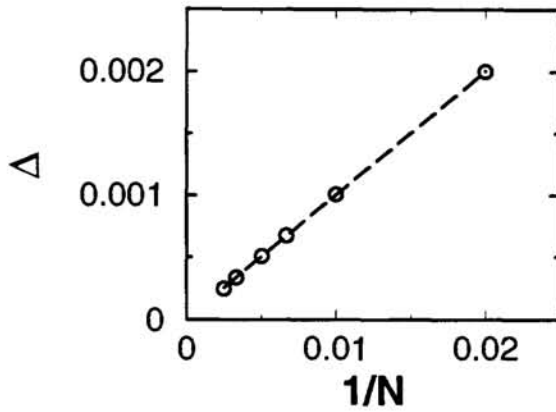

Figure 2: Scaling of $\Delta$ for a perceptron with two Fourier components, (eq. 9), with $a_i = 1$, $K_1 = 3$, $\phi_1 = 0.121$, $K_2 = 7$, $\phi_2 = 0$, $W_0 = 0.3$. The dashed line is a linear fit of $0.1/N$, $N = 50, \ldots, 400$.

Applying $\mathcal{T}$ $\tau$ - times to an initial state vector results in a vector whose second largest component is of order:

$$|\lambda_2|^\tau = (1 - \Delta)^\tau = \exp\{\tau \log(1 - \Delta)\} \overset{\Delta \ll 1}{\approx} \exp\{-\tau\Delta\} \qquad (13)$$

therefore we can define the characteristic relaxation time in the vicinity of an attractor to be $\tau = \Delta^{-1}$. [1]

We have analysed eq. (12) numerically for various cases of $c_i$, e.g. $W_i$ composed of one or two Fourier components. In all the cases $\beta$ was chosen to be the minimal $\beta_c$ to ensure that the linearized form is valid. We found that $\Delta \sim 1/N$. Figure 2 depicts one example of two Fourier components. Next, we have simulated the network and measured the average time ($\overline{\tau^s}$) it takes to flow into an attractor starting from an arbitrary initial condition. The following simulations support the analytical result ($\tau \sim N$) for general (random) weights and high gain ($\beta$) value as well. The threshold we apply for the decision whether the flow is already close enough to the attractor is the ratio between the component with the largest power in the spectrum and the total power spectrum of the current state ($\overline{S}^t$), which should exceed 0.95. The results presented in Figure 3 are an average over 100 samples started from random initial condition. The weights are taken at random, however we add a dominant Fourier component with no phase to control the bifurcation point more easily. This component has an amplitude which is about twice the other components to make sure that its bifurcation point is the smallest. We observe a clear linear relation between this time and $N$ ($\overline{\tau^s} \sim N$). The slope depends on the actual values of the weights, however the power law scaling does not change.

On general principles, we expect the analytically derived scaling law for $\Delta$ to be valid even beyond the linear regime. Indeed the numerical simulations (Figure 3) support this conjecture.

## 3 Multilayer networks

For simplicity, we restrict the present analysis to a multilayer network (MLN) with $N$ inputs, $H$ hidden units and a single linear output, however this restriction can be removed, e.g. nonlinear output and more hidden layers. The units in the hidden layer are the perceptrons discussed above and the output is given by:

$$S_{out} = \sum_{m=1}^{H} \tanh\left[\beta\left(\sum_{j=1}^{N}(W_j^m + W_0^m)S_j^t\right)\right] \qquad (14)$$

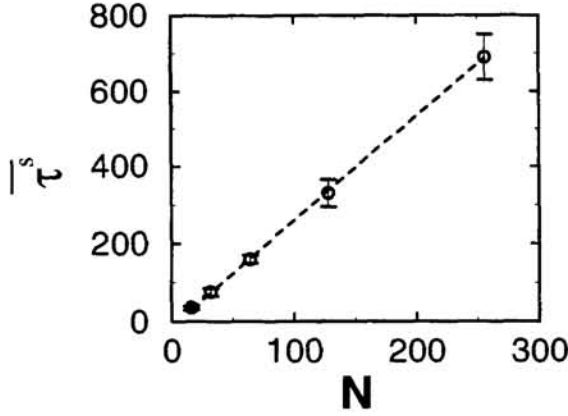

Figure 3: Scaling of $\overline{\tau^s}$ for random weights with a dominant component at $K = 7$, $\phi = 0$, $a = 1$; All other amplitudes are randomly taken between $(0, 0.5)$ and the phases are random as well. $\beta = 3.2/N$. The dashed line is a linear fit of $cN$, $c = 2.73 \pm 0.03$. $N = 16, \ldots, 256$.

The dynamic rule is defined by eq. (1). First consider the case where the weights of each hidden unit are of the form described by eq. (6), i.e. each hidden unit has only one (possibly biased) Fourier component:

$$W_j^m = a_m \cos(2\pi K_m j/N - \pi\phi_m) \; ; \quad W_0^m = b_m \qquad m = 1, \ldots, H. \qquad (15)$$

Following a similar treatment as for the perceptron, the stationary solution is a combination of the perceptron-like solution:

$$S^l = \sum_{m=1}^{H} \tanh\left[A_m(\beta) \cos(2\pi(K_m - \phi_m)l/N) + B_m(\beta)\right] \qquad (16)$$

The variables $A_m, B_m$ are the solution of the self consistent coupled equations, however by contrast with the single perceptron, each hidden unit operates independently and can potentially develop an attractor of the type described in section 1. The number of attractors depends on $\beta$ with a maximum of $H$ attractors. The number of non-zero $A_m$'s defines the attractor's dimension in the generic case of irrational $\phi$'s associated with them. If different units do not share Fourier components with a common divisor or harmonics of one another, it is easy to define the quantitative result, otherwise, one has to analyse the coupled equations more carefully to find the exact value of the variables. Nevertheless, each hidden unit exhibits only a single highly dominant component ($A \neq 0$ or $B \neq 0$).

Generalization of this result to more than a single biased Fourier component is straightforward. Each vector is of the form described in eq. (9) plus an index for the hidden unit. The solution is a combination of the general perceptron solution, eq. (10). This solution is much more involved and the coupled equations are complicated but careful study of them reveals the same conclusion, namely each hidden unit possess a single dominant Fourier component (possibly with several other much smaller due to the other components in the vector). As the gain parameter $\beta$ becomes larger, more components becomes available and the number of possible attractors increases. For a very large value it is possible that higher harmonics from different hidden units might interfere and complicate considerably the solution. Still, one can trace the origin of this behavior by close inspection of the fields in each hidden unit.

We have also measured the relaxation time associated with MLN's in simulations. The preliminary results are similar to the perceptron, i.e. $\overline{\tau^s} \sim N$ but the constant prefactor is larger when the weights consist of more Fourier components.

## 4 Discussion

Neural networks were proved to be universal approximators (e.g. Hornik, 1991), hence they are capable of approximating the prediction function of the delay coordinate vector. The conclusion should be that prediction is indeed possible. This observation holds only for short times in general. As we have shown, long time predictions are governed by the attractor dynamics described above. The results point out the conclusion that the asymptotic behavior for this networks is dictated by the architecture and not by the details of the weights. Moreover, the attractor dimension of the asymptotic sequence is typically bounded by the number of hidden units in the first layer (assuming the network does not contain internal delays). To prevent any misunderstanding we note again that this result refers to the asymptotic behavior although the short term sequence can approximate a very complicated attractor.

The main result can be interpreted as follows. Since the network is able to approximate the prediction function, the initial condition is followed by reasonable predictions which are the mappings from the vicinity of the original manifold created by the network. As the trajectory evolves, it flows to one of the attractors described above and the predictions are no longer valid. In other words, the initial combination of solutions described in eq. (10) or its extension to MLN (with an arbitrary number of non-zero variables, $A$'s or $B$'s) serves as the approximate mapping. Evolution of this approximation is manifested in the variables of the solution, which eventually are attracted to a stable attractor (in the non-chaotic regime). The time scale for the transition is given by the relaxation time developed above.

The formal study can be applied for practical purposes in two ways. First, taking into account this behavior by probing the generated sequence and looking for its indications. One such indication is stationarity of the power spectrum. Second, one can incorporate ideas from local linear models in the reconstructed space to restrict the inputs in such a way that they always remain in the vicinity of the original manifold (Sauer, in Weigand and Gershenfeld, 1994).

**Acknowledgments**

This research has been supported by the Israel Science Foundation.

**References**

Weigand A. S. and Gershenfeld N. A. ; *Time Series Prediction*, Addison-Wesley, Reading, MA, 1994.

E. Eisenstein, I. Kanter, D. A. Kessler and W. Kinzel ; Generation and prediction of time series by a neural network, Phys. Rev. Lett. **74**, 6 (1995).

Waibel A., Hanazawa T., Hinton G., Shikano K. and Lang K.; Phoneme recognition using TDNN, IEEE Trans. Acoust., Speech & Signal Proc. **37(3)**, (1989).

Takens F., Detecting strange attractors in turbulence, in Lecture notes in mathematics vol. 898, Springer-Verlag, 1981.

T. Sauer, J. A. Yorke and M. Casdagli; Embedology, J. Stat. Phys. **65(3)**, (1991).

Ralston A. and Rabinowitz P. ; *A first course in numerical analysis*, McGraw-Hill, 1978.

K. Hornik; Approximation capabilities of multilayer feed forward networks, Neural Networks 4, (1991).

## Footnotes

[1]Note that if one demand the L.H.S. of eq. (13) to be of $\mathcal{O}(\Delta)$, then $\tau \sim \Delta^{-1}\log(\Delta^{-1})$.
